# Temporal Adaptation
# in a
# Silicon Auditory Nerve

**John Lazzaro**

CS Division
UC Berkeley
571 Evans Hall
Berkeley, CA 94720

## Abstract

Many auditory theorists consider the temporal adaptation of the auditory nerve a key aspect of speech coding in the auditory periphery. Experiments with models of auditory localization and pitch perception also suggest temporal adaptation is an important element of practical auditory processing. I have designed, fabricated, and successfully tested an analog integrated circuit that models many aspects of auditory nerve response, including temporal adaptation.

## 1. INTRODUCTION

We are modeling known and proposed auditory structures in the brain using analog VLSI circuits, with the goal of making contributions both to engineering practice and biological understanding. Computational neuroscience involves modeling biology at many levels of abstraction. The first silicon auditory models were constructed at a fairly high level of abstraction (Lyon and Mead, 1988; Lazzaro and Mead, 1989ab; Mead et al., 1991; Lyon, 1991). The functional limitations of these silicon systems have prompted a new generation of auditory neural circuits designed at a lower level of abstraction (Watts et al., 1991; Liu et al., 1991).

The silicon model of auditory nerve response models sensory transduction and spike generation in the auditory periphery at a high level of abstraction (Lazzaro and Mead, 1989c); this circuit is a component in silicon models of auditory localization, pitch perception, and spectral shape enhancement (Lazzaro and Mead, 1989ab; Lazzaro, 1991a). Among other limitations, this circuit does not model the short-term temporal adaptation of the auditory nerve. Many auditory theorists consider the temporal adaptation of the auditory nerve a key aspect of speech coding in the auditory periphery (Delgutte and Kiang, 1984). From the engineering perspective, the pitch perception and auditory localization chips perform well with sustained sounds as input; temporal adaptation in the silicon auditory nerve should improve performance for transient sounds.

I have designed, fabricated, and tested an integrated circuit that models the temporal adaptation of spiral ganglion neurons in the auditory periphery. The circuit receives an analog voltage input, corresponding to the signal at an output tap of a silicon cochlea, and produces fixed-width, fixed-height pulses that are correlates to the action potentials of an auditory nerve fiber. I have also fabricated and tested an integrated circuit that combines an array of these neurons with a silicon cochlea (Lyon and Mead, 1988); this design is a silicon model of auditory nerve response. Both circuits were fabricated using the Orbit double polysilicon n-well $2\mu m$ process.

## 2. TEMPORAL ADAPTATION

Figure 1 shows data from the temporal adaptation circuit; the data in this figure was taken by connecting signals directly to the inner hair cell circuit input, bypassing silicon cochlea processing. In (a), we apply a 1 kHz pure tone burst of 20ms in duration to the input of the hair cell circuit (top trace), and see an adapting sequence of spikes as the output (middle trace). If this tone burst in repeated at 80ms intervals, each response in unique; by averaging the responses to 64 consecutive tone bursts (bottom trace), we see the envelope of the temporal adaptation superimposed on the cycle-by-cycle phase-locking of the spike train. These behaviors qualitatively match biological experiments (Kiang et al., 1965).

In biological auditory nerve fibers, cycle-by-cycle phase locking ceases for auditory fibers tuned to sufficiently high frequencies, but the temporal adaptation property remains. In the silicon spiral ganglion neuron, a 10kHz pure tone burst fails to elicit phase-locking (Figure 1(b), trace identities as in (a)). Temporal adaptation remains, however, qualitatively matching biological experiments (Kiang et al., 1965).

To compare this data with the previous generation of silicon auditory nerve circuits, we set the control parameters of the new spiral ganglion model to eliminate temporal adaptation. Figure 1(c) shows the 1 kHz tone burst response (trace identities as in (a)). Phase locking occurs without temporal adaptation. The uneven response of the averaged spike outputs is due to beat frequencies between the input tone frequency and the output spike rate; in practice, the circuit noise of the silicon cochleas adds random variation to the auditory input and smooths this response (Lazzaro and Mead, 1989c).

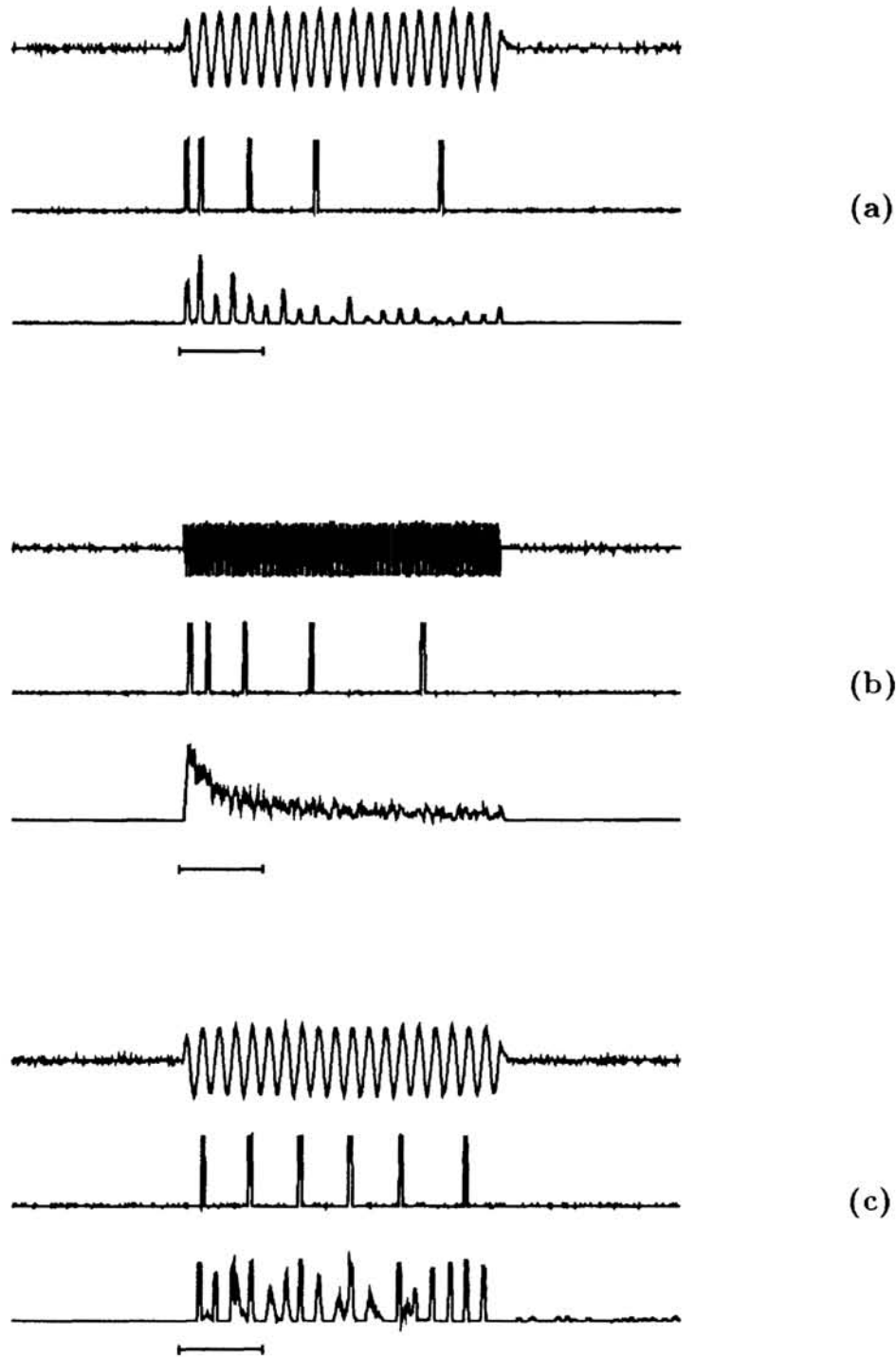

**Figure 1.** Responses of test chip to pure tone bursts. Horizontal axis is time for all plots, all horizontal rules measure 5 ms. **(a)** Chip response to a 1 kHz, 20 ms tone burst. Top trace shows tone burst input, middle trace shows a sample response from the chip, bottom trace shows averaged output of 64 responses to tone bursts. Averaged response shows both temporal adaptation and phase locking. **(b)** Chip response to a 10 kHz, 20 ms tone burst. Trace identifications identical to (a). Response shows temporal adaptation without phase locking. **(c)** Chip response to a 1 kHz, 20 ms tone burst, with adaptation circuitry disabled. Trace identifications identical to (a). Response shows phase locking without temporal adaptation.

## 3. CIRCUIT DESIGN

Figure 2 shows a block diagram of the model. The circuits modeling inner hair cell transduction remain unchanged from the original model (Lazzaro and Mead, 1989c), and are shown as a single box. This box performs time differentiation, nonlinear compression and half-wave rectification on the input waveform $V_i$, producing a unidirectional current waveform as output. The dependent current source represents this processed signal.

The axon hillock circuit (Mead, 1989), drawn as a box marked with a pulse, converts this current signal into a series of fixed-width, fixed height spikes; $V_o$ is the output of the model. The current signal is connected to the pulse generator using a novel current mirror circuit, that serves as the control element to regulate temporal adaptation. This current mirror circuit has an additional high impedance input, $V_a$, that exponentially scales the current entering the axon hillock circuit (the current mirror operates in the subthreshold region). The adaptation capacitor $C_a$ is associated with the control voltage $V_a$.

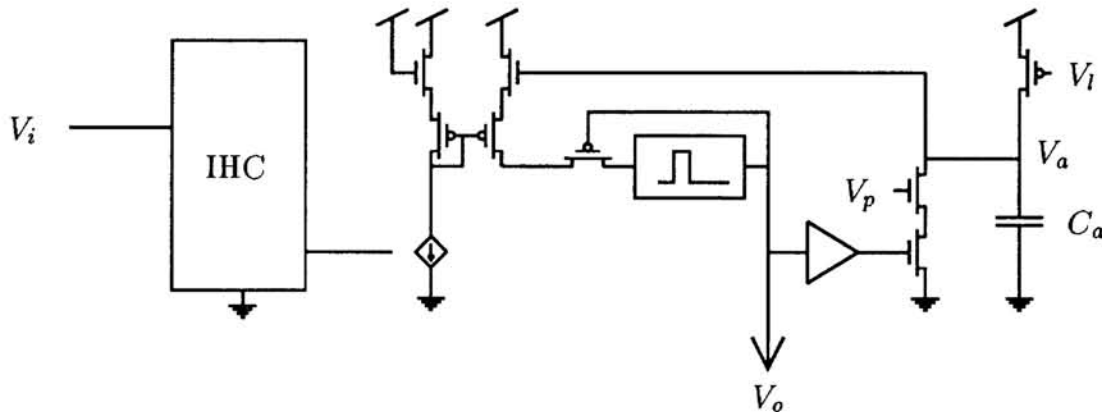

**Figure 2.** Circuit schematic of the enhanced silicon model of auditory nerve response. The circuit converts the analog voltage input $V_i$ into the pulse train $V_o$; control voltages $V_l$ and $V_p$ control the temporal adaptation of state variable $V_a$ on capacitor $C_a$. See text for details.

$C_a$ is constantly charged by the PFET transistor associated with control voltage $V_l$, and is discharged during every pulse output of the axon hillock circuit, by an amount set by the control voltage $V_p$. During periods with no input signal, $V_a$ is charged to $V_{dd}$, and the current mirror is set to deliver maximum current with the onset of an input signal. If an input signal occurs and neuron activity begins, the capacitor $V_a$ is discharged with every spike, degrading the output of the current mirror. In this way, temporal adaptation occurs, with characteristics determined by $V_p$ and $V_l$.

The nonlinear differential equations for this adaptation circuit are similar to the equations governing the adaptive baroreceptor circuit (Lazzaro et al., 1991); the publication describing this circuit includes an analysis deriving a recurrence relation that describes the pulse output of the circuit given a step input.

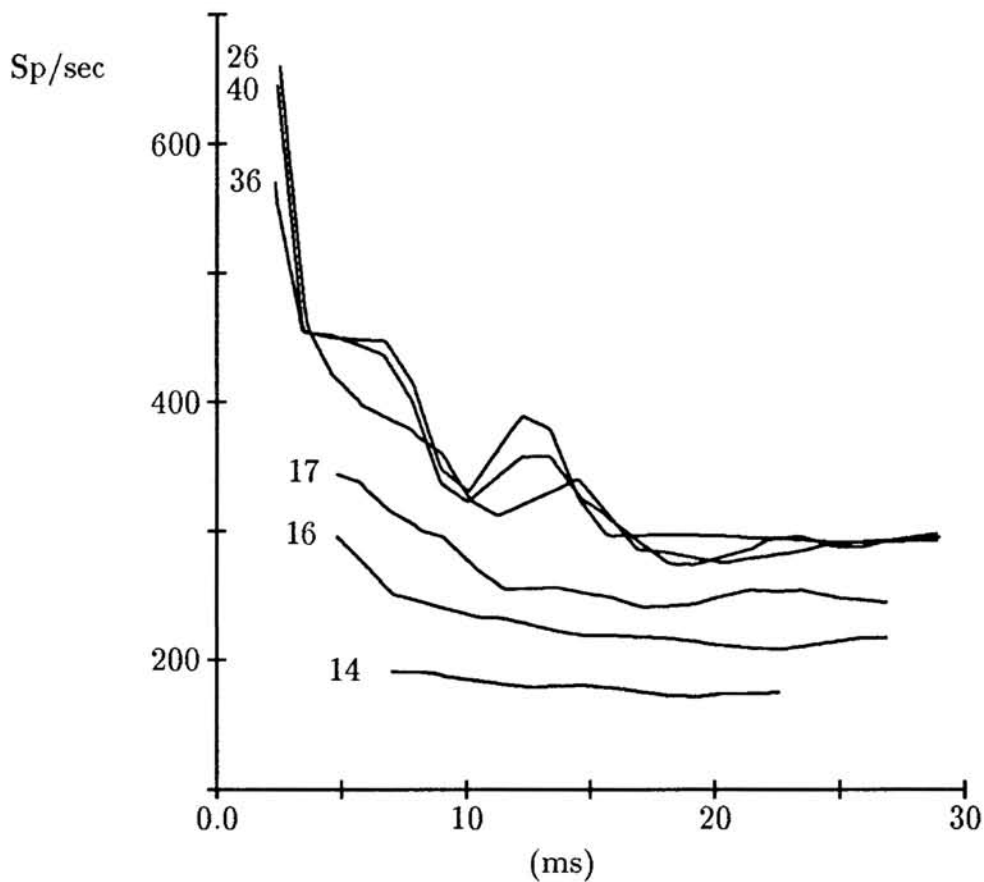

**Figure 3.** Instantaneous firing rate of the adaptive neuron, as a function of time; tone burst begins at 0 ms. Each curve is marked with the amplitude of presented tone burst, in dB. Tone burst frequency is 1Khz.

## 4. DATA ANALYSIS

The experiment shown in Figure 1(a) was repeated for tone bursts of different amplitudes; this data set was used to produce several standard measures of adaptive response (Hewitt and Meddis, 1991). The integrated auditory nerve circuit was used for this set of experiments. Data was taken from an adaptive auditory nerve output that had a best frequency of 1 Khz.; the frequency of all tone bursts was also 1 Khz.

Figure 3 shows the instantaneous firing rate of the auditory nerve output as a function of time, for tone bursts of different amplitudes. Adaptation was more pronounced for more intense sounds. This difference is also seen in Figure 4. In this figure, instantaneous firing rate is plotted as a function of amplitude, both at response onset and after full adaptation.

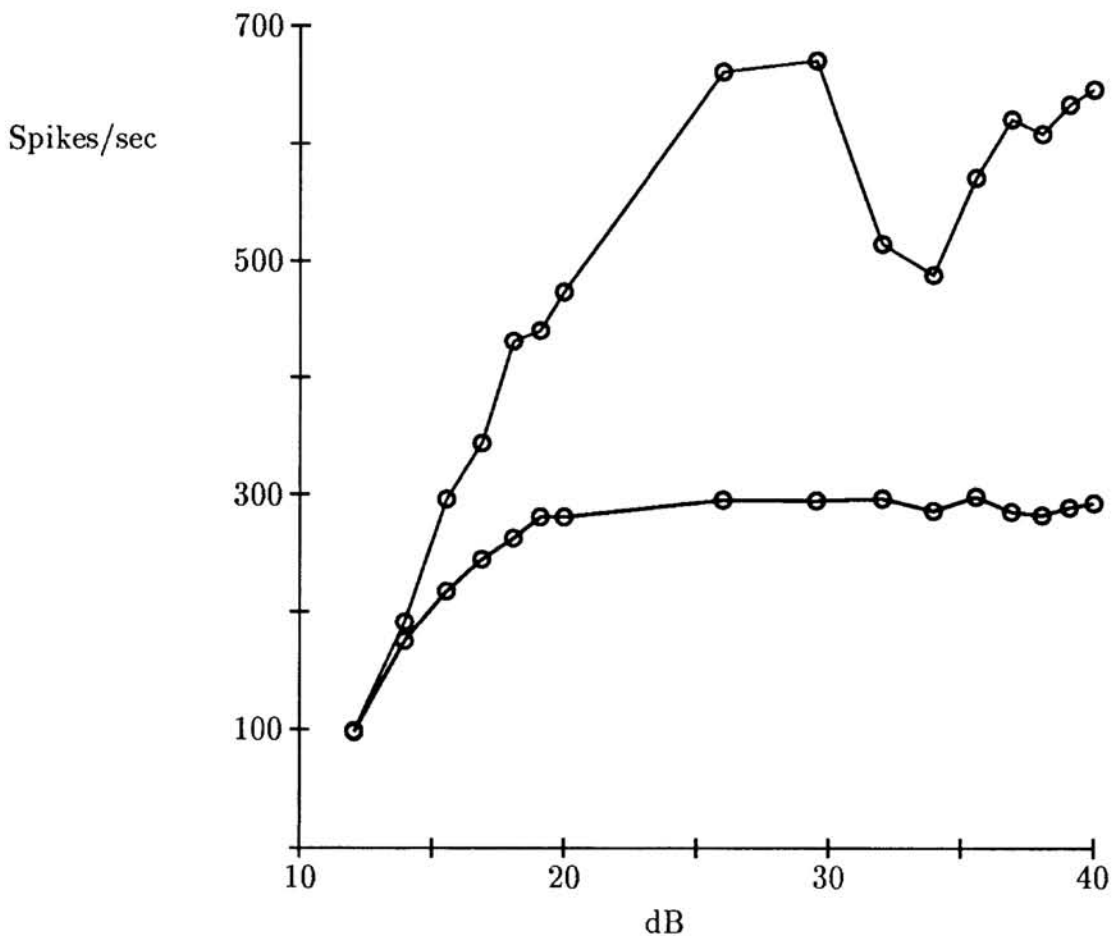

**Figure 4.** Instantaneous firing rate of the adaptive neuron, as a function of amplitude (in dB). Top curve is firing rate at onset of response, bottom curve is firing rate after adaptation. Tone burst frequency is 1Khz.

Figure 4 shows that the instantaneous spike rate saturates at moderate intensity after full adaptation; at these moderate intensities, however, the onset instantaneous spike rate continues to encode intensity. Figure 4 shows a non-monotonicity at high intensities in the onset response; this undesired non-monotonicity is a result of the undesired saturation of the silicon cochlea circuit (Lazzaro, 1991b).

## 5. CONCLUSION

This circuit improves the silicon model of auditory response, by adding temporal adaptation. We expect this improvement to enhance existing architectures for auditory localization and pitch perception, and aid the creation of new circuits for speech processing.

### Acknowledgements

Thanks to K. Johnson of CU Boulder and J. Wawrzynek of UC Berkeley for hosting this research in their laboratories. I also thank the Caltech auditory research community, specifically C. Mead, D. Lyon, M. Konishi, L. Watts, M. Godfrey, and X. Arreguit. This work was funded by the National Science Foundation.

### References

Delgutte, B., and Kiang, Y. S. (1984). Speech coding in the auditory nerve I-V. *J. Acoust. Soc. Am* **75**:3, 866–918.

Hewitt, M. J. and Meddis, R. (1991). An evaluation of eight computer models of mammalian inner hair-cell function. *J. Acoust. Soc. Am* **90**:2, 904.

Kiang, N. Y.-s, Watenabe, T., Thomas, E.C., and Clark, L.F. (1965). *Discharge Patterns of Single Fibers in the Cat's Auditory Nerve*. Cambridge, MA: M.I.T Press.

Lazzaro, J. and Mead, C. (1989a). A silicon model of auditory localization. *Neural Computation* **1**: 41–70.

Lazzaro, J. and Mead, C. (1989b). Silicon modeling of pitch perception. *Proceedings National Academy of Sciences* **86**: 9597–9601.

Lazzaro, J. and Mead, C. (1989c). Circuit models of sensory transduction in the cochlea. In Mead, C. and Ismail, M. (eds), *Analog VLSI Implementations of Neural Networks*. Norwell, MA: Kluwer Academic Publishers, pp. 85-101.

Lazzaro, J. P. (1991a). A silicon model of an auditory neural representation of spectral shape. *IEEE Journal Solid State Circuits* **26**: 772–777.

Lazzaro, J. P. (1991b). Biologically-based auditory signal processing in analog VLSI. *IEEE Asilomar Conference on Signals, Systems, and Computers*.

Lazzaro, J. P., Schwaber, J., and Rogers, W. (1991). Silicon baroreceptors: modeling cardiovascular pressure transduction in analog VLSI. In Sequin, C. (ed), *Ad-*

*vanced Research in VLSI, Proceedings of the 1991 Santa Cruz Conference,* Cambridge, MA: MIT Press, pp. 163–177.

Liu, W., Andreou, A., and Goldstein, M. (1991). Analog VLSI implementation of an auditory periphery model. *25 Annual Conference on Information Sciences and Systems,* Baltimore, MD, 1991.

Lyon, R. and Mead, C. (1988). An analog electronic cochlea. *IEEE Trans. Acoust., Speech, Signal Processing* 36: 1119–1134.

Lyon, R. (1991). CCD correlators for auditory models. *IEEE Asilomar Conference on Signals, Systems, and Computers.*

Mead, C. A., Arreguit, X., Lazzaro, J. P. (1991) Analog VLSI models of binaural hearing. *IEEE Journal of Neural Networks,* 2: 230–236.

Mead, C. A. (1989). *Analog VLSI and Neural Systems.* Reading, MA: Addison-Wesley.

Watts, L., Lyon, R., and Mead, C. (1991). A bidirectional analog VLSI cochlear model. In Sequin, C. (ed), *Advanced Research in VLSI, Proceedings of the 1991 Santa Cruz Conference,* Cambridge, MA: MIT Press, pp. 153–163.